# Robot Docking using Mixtures of Gaussians

Matthew Williamson*    Roderick Murray-Smith[†]    Volker Hansen[‡]

## Abstract

This paper applies the *Mixture of Gaussians* probabilistic model, combined with Expectation Maximization optimization to the task of summarizing three dimensional range data for a mobile robot. This provides a flexible way of dealing with uncertainties in sensor information, and allows the introduction of prior knowledge into low-level perception modules. Problems with the basic approach were solved in several ways: the mixture of Gaussians was reparameterized to reflect the types of objects expected in the scene, and priors on model parameters were included in the optimization process. Both approaches force the optimization to find 'interesting' objects, given the sensor and object characteristics. A higher level classifier was used to interpret the results provided by the model, and to reject spurious solutions.

## 1 Introduction

This paper concerns an application of the *Mixture of Gaussians* (MoG) probabilistic model (Titterington et al., 1985) for a robot docking application. We use the Expectation-Maximization (EM) approach (Dempster et al., 1977) to fit Gaussian sub-models to a sparse 3d representation of the robot's environment, finding walls, boxes, etc.. We have modified the MoG formulation in three ways to incorporate prior knowledge about the task, and the sensor characteristics: the parameters of the Gaussians are recast to constrain how they fit the data, priors on these parameters are calculated and incorporated into the EM algorithm, and a higher level processing stage is included which interprets the fit of the Gaussians on the data, detects misclassifications, and providing prior information to guide the model-fitting.

The robot is equipped with a LIDAR 3d laser range-finder (PIAP, 1995) which it uses to identify possible docking objects. The range-finder calculates the time of flight for a light pulse reflected off objects in the scene. The particular LIDAR used is not very powerful, making objects with poor reflectance (e.g., dark, shiny, or surfaces not perpendicular to the

*Corresponding author: MIT AI Lab, Cambridge, MA, USA. matt@ai.mit.edu
[†]Dept. of Mathematical Modelling, Technical University of Denmark. rod@imm.dtu.dk
[‡]DaimlerChrysler, Alt-Moabit 96a, Berlin, Germany. hansen@dbag.bln.daimlerbenz.com

laser beam) invisible. The scan pattern is also very sparse, especially in the vertical direction, as shown in the scan of a wall in Figure 1. However, if an object is detected, the range returned is accurate (±1-2cm). When the range data is plotted in Cartesian space it forms a number of sparse clusters, leading naturally to the use of MoG clustering algorithms to make sense of the scene. While the Gaussian assumption is not an ideal model of the data, the generality of MoG, and its ease of implementation and analysis motivated its use over a more specialized approach. The sparse nature of the data inspired the modifications to the MoG formulation described in this paper.

Model-based object recognition from dense range images has been widely reported (see (Arman and Aggarwal, 1993) for a review), but is not relevant in this case given the sparseness of the data. Denser range images could be collected by combining multiple scans, but the poor visibility of the sensor hampers the application of these techniques. The advantage of the MoG technique is that the segmentation is "soft", and perception proceeds iteratively during learning. This is especially useful for mobile robots where evidence accumulates over time, and the allocation of attention is time and state-dependent. The EM algorithm is useful since it is guaranteed to converge to a local maximum.

The following sections of the paper describe the re-parameterization of the Gaussians to model plane-like clusters, the formulation of the priors, and the higher level processing which interprets the clustered data in order to both move the robot and provide prior information to the model-fitting algorithm.

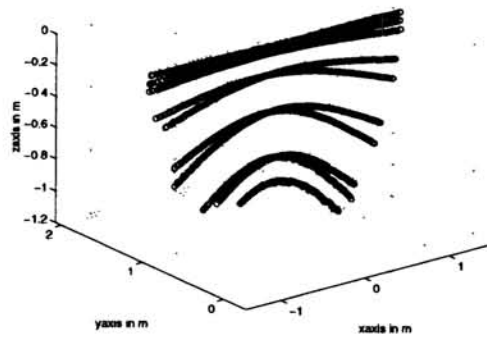

Figure 1: Plot showing data from a LIDAR scan of a wall, plotted in Cartesian space. The robot is located at the origin, with the $y$ axis pointing forward, $x$ to the right, and $z$ up. The sparse scan pattern is visible, as well as the visibility constraint: the wall extends beyond where the scan ends, but is invisible to the LIDAR due to the orientation of the wall

## 2  Mixture of Gaussians model

The range-finder returns a set of data, each of which is a position in Cartesian space $\mathbf{x}_i = (x_i, y_i, z_i)$. The complete set of data $D = \{\mathbf{x}_1 \ldots \mathbf{x}_N\}$ is modeled as being generated by a mixture density

$$P(\mathbf{x}_n) = \sum_{i=1}^{M} P(\mathbf{x}_n|i, \mu_i, \Sigma_i, \pi_i)P(i),$$

where we use a Gaussian as the sub-model, with mean $\mu_i$, variance $\Sigma_i$ and weight $\pi_i$, which makes the probability of a particular data point:

$$P(\mathbf{x}_n|\mu, \Sigma, \pi) = \sum_{i=1}^{M} \frac{\pi_i}{(2\pi)^{3/2}|\Sigma_i|^{1/2}} \exp\left(-\frac{1}{2}(\mathbf{x}_n - \mu_i)^T \Sigma_i^{-1}(\mathbf{x}_n - \mu_i)\right)$$

Given a set of data $D$, the most likely set of parameters is found using the EM algorithm. This algorithm has a number of advantages, such as guaranteed convergence to a local minimum, and efficient computational performance.

In 3D Cartesian space, the Gaussian sub-models form ellipsoids, where the size and orientation are determined by the covariance matrix $\Sigma_i$. In the general case, the EM algorithm can be used to learn all the parameters of $\Sigma_i$. The sparseness of the LIDAR data makes this parameterization inappropriate, as various odd collections of points could be clustered together. By changing the parameterization of $\Sigma_i$ to better model plane-like structures, the system can be improved. The reparameterization is most readily expressed in terms of the eigenvalues $\Lambda_i$ and eigenvectors $V_i$ of the covariance matrix $\Sigma_i = V_i \Lambda_i V_i^{-1}$.

The covariance matrix of a normal approximation to a plane-like vertical structure will have a large eigenvalue in the $z$ direction, and in the $x$–$y$ plane one large and one small eigenvalue. Since $\Sigma_i$ is symmetrical, the eigenvectors are orthogonal, $V_i^{-1} = V_i^T = V_i$, and $\Sigma_i$ can be written:

$$\Sigma_i = \begin{pmatrix} \sin\theta_i & \cos\theta_i & 0 \\ \cos\theta_i & -\sin\theta_i & 0 \\ 0 & 0 & 1 \end{pmatrix} \begin{pmatrix} a_i & 0 & 0 \\ 0 & \gamma a_i & 0 \\ 0 & 0 & b_i \end{pmatrix} \begin{pmatrix} \sin\theta_i & \cos\theta_i & 0 \\ \cos\theta_i & -\sin\theta_i & 0 \\ 0 & 0 & 1 \end{pmatrix},$$

where $\theta_i$ is the angle of orientation of the $i$th sub-model in the $x$–$y$ plane, $a_i$ scales the cluster in the $x$ and $y$ directions, and $b_i$ scales in the $z$ direction. The constant $\gamma$ controls the aspect ratio of the ellipsoid in the $x$–$y$ plane.[1]

The optimal values of these parameters $(a, b)$ are found using EM, first calculating the probability that data point $\mathbf{x}_n$ is modeled by Gaussian $i$, $(h_{in})$ for every data point $\mathbf{x}_n$ and every Gaussian $i$,

$$h_{in} = \frac{\pi_i |\Sigma_i|^{-1/2} \exp\left(-\frac{1}{2}(\mathbf{x}_n - \mu_i)^T \Sigma_i^{-1}(\mathbf{x}_n - \mu_i)\right)}{\sum_{i=1}^M \pi_i |\Sigma_i|^{-1/2} \exp\left(-\frac{1}{2}(\mathbf{x}_n - \mu_i)^T \Sigma_i^{-1}(\mathbf{x}_n - \mu_i)\right)}.$$

This "responsibility" is then used as a weighting for the updates to the other parameters,

$$\hat{\mu}_i = \frac{\sum_n h_{in}\mathbf{x}_n}{\sum_n h_{in}}, \quad \hat{\theta}_i = \frac{1}{2}\tan^{-1}\left(\frac{2\sum_n h_{in}(\mathbf{x}_{n1} - \mu_{i1})(\mathbf{x}_{n2} - \mu_{i2})}{\sum_n h_{in}[(\mathbf{x}_{n1} - \mu_{i1})^2 - (\mathbf{x}_{n2} - \mu_{i2})^2]}\right)$$

$$\zeta = (\gamma - 1)((\mathbf{x}_{n1} - \mu_{i1})\sin\theta + (\mathbf{x}_{n2} - \mu_{i2})\cos\theta)^2 + (\mathbf{x}_{n1} - \mu_{i1})^2 + (\mathbf{x}_{n2} - \mu_{i2})^2$$

$$\hat{a}_i = \frac{\sum_n h_{in}\zeta}{2\gamma \sum_n h_{in}}, \quad \hat{b}_i = \frac{\sum_n h_{in}(\mathbf{x}_{n3} - \mu_{n3})^2}{\sum_n h_{in}},$$

where $\mathbf{x}_{n1}$ is the first element of $\mathbf{x}_n$ etc. and $\zeta$ corresponds to the projection of the data into the plane of the cluster. It is important to update the means $\mu_i$ first, and use the new values to update the other parameters.[2] Figure 2 shows a typical model response on real LIDAR data.

## 2.1 Practicalities of application, and results

Starting values for the model parameters are important, as EM is only guaranteed to find a local optimum. The Gaussian mixture components are initialized with a large covariance, allowing them to pick up data and move to the correct positions. We found that initializing the means $\mu_i$ to random data points, rather than randomly in the input space, tended to

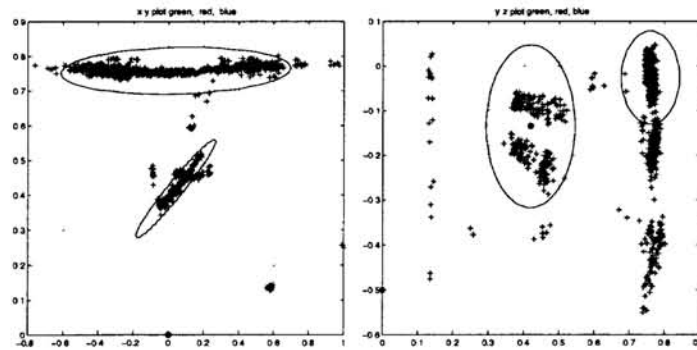

Figure 2: Example of clustering of the 3d data points. The left hand graph shows the view from above (the $x$–$y$ plane), and the right graph shows the view from the side (the $y$–$z$ plane), with the robot positioned at the origin. The scene shows a box at an oblique angle, with a wall behind. The extent of the plane-like Gaussian sub-models is illustrated using the ellipses, which are drawn at a probability of 0.5.

work better, especially given the sensor characteristics—if the LIDAR returned a range measurement, it was likely to be part of an interesting object.

Despite the accuracy of measurement, there are still outlying data points, and it is impossible to fully segment the data into separate objects. One simple solution we found was to define a "junk" Gaussian. This is a sub-model placed in the center of the data, with a large covariance $\Sigma$. This Gaussian then becomes responsible for the outliers in the data (i.e. sparsely distributed data over the whole scene, none of which are associated with a specific object), allowing the object-modeling Gaussians to work undistracted.

The use of EM with the $a, b, \theta$ parameterization found and represented plane-like data clusters better than models where all the elements of the covariance matrix were free to adapt. It also tended to converge faster, probably due to the reduced numbers of parameters in the covariance matrix (3 as opposed to 6). Although the algorithm is constrained to find planes, the parameterization was flexible enough to model other objects such as thin vertical lines (say from a table leg). The only problem with the algorithm was that it occasionally found poor local minimum solutions, such as illustrated in Figure 3. This is a common problem with least squares based clustering methods (Duda and Hart, 1973).

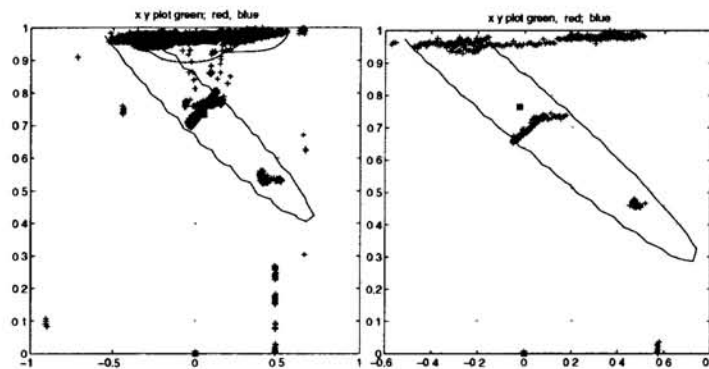

Figure 3: Two examples of 'undesirable' local minimum solutions found by EM. Both graphs show the top view of a scene of a box in front of a wall. The algorithm has incorrectly clustered the box with the left hand side of the wall.

## 3 Incorporating prior information

As well as reformulating the Gaussian models to suit our application, we also incorporated prior knowledge on the parameters of the sub-models. Sensor characteristics are often well-defined, and it makes sense to use these as early as possible in perception, rather than dealing with their side-effects at higher levels of reasoning. Here, e.g., the visibility constraint, by which only planes which are almost perpendicular to the lidar rays are visible, could be included by writing $P(\mathbf{x}_n) = \sum_{i=1}^{M} P(\mathbf{x}_n|i, \beta_i)P(i)P(visible|\beta_i)$, the updates could be recalculated, and the feature immediately brought into the modeling process. In addition, prior knowledge about the locations and sizes of objects, maybe from other sensors, can be used to influence the modeling procedure. This allows the sensor to make better use of the sparse data.

For a model with parameters $\beta$ and data $D$, Bayes rule gives:

$$P(\beta|D) = \frac{P(\beta)}{P(D)} \prod P(\mathbf{x}_n|\beta).$$

Normally the logarithm of this is taken, to give the log-likelihood, which in the case of mixtures of Gaussians is

$$L(D|\beta) = \log(p(\{\mu_i, \pi_i, a_i, b_i, \theta_i\})) - \log(p(D)) + \sum_n \log \sum_i p(\mathbf{x}_n|i, \mu_i, \pi_i, a_i, b_i, \theta_i)$$

To include the parameter priors in the EM algorithm, distributions for the different parameters are chosen, then the log-likelihood is differentiated as usual to find the updates to the parameters (McMichael, 1995). The calculations are simplified if the priors on all the parameters are assumed to be independent, $p(\{\mu_i, \pi_i, a_i, b_i, \theta_i\}) = \prod_i p(\mu_i)p(\pi_i)p(a_i)p(b_i)p(\theta_i)$.

The exact form of the prior distributions varies for different parameters, both to capture different behavior and for ease of implementation. For the element means ($\mu_i$), a flat distribution over the data is used, specifying that the means should be among the data points. For the element weights, a multinomial Dirichlet prior can be used, $p(\pi_i|\alpha) = \frac{\Gamma(\alpha+M)}{\Gamma(\alpha+1)^M} \prod_{i=1}^{M} \pi_i^{\alpha}$. When the hyperparameter $\alpha > 0$, the algorithm favours weights around $1/M$, and when $-1 < \alpha < 0$, weights close to 0 or 1.[3] The expected value of $a_i$ (written as $\overline{a_i}$) can be encoded using a truncated inverse exponential prior (McMichael, 1995), setting $p(a_i|\overline{a_i}) = K \exp(-\overline{a_i}/(2a_i))$, where $K$ is a normalizing factor.[4] The prior for $b_i$ has the same form. Priors for $\theta_i$ were not used, but could be useful to capture the visibility constraint. Given these distributions, the updates to the parameters become

$$\hat{\mu}_i = \frac{\sum_n h_{in}\mathbf{x}_n}{\sum_n h_{in}}, \qquad \hat{\pi}_i = \frac{\sum_n h_{in} + \alpha}{\sum_n \sum_j h_{jn} + \alpha}$$

$$\hat{a}_i = \frac{\sum_n h_{in}\zeta/\gamma + \overline{a_i}}{2\sum_n h_{in}}, \qquad \hat{b}_i = \frac{\sum_n h_{in}(\mathbf{x}_{n3} - \mu_{n3})^2 + \overline{b_i}}{\sum_n h_{in}}.$$

The update for $\mu_i$ is the same as before, the prior having no effect. The update for $a_i$ and $b_i$ forces them to be near $\overline{a_i}$ and $\overline{b_i}$, and the update for $\pi_i$ is affected by the hyperparameter $\alpha$.

The priors on $a_i$ and $b_i$ had noticeable effects on the models obtained. Figure 4 shows the results from two fits, starting from identical initial conditions. By adjusting the size of the prior, the algorithm can be guided into finding different sized clusters. Large values of the prior are shown here to demonstrate its effect.

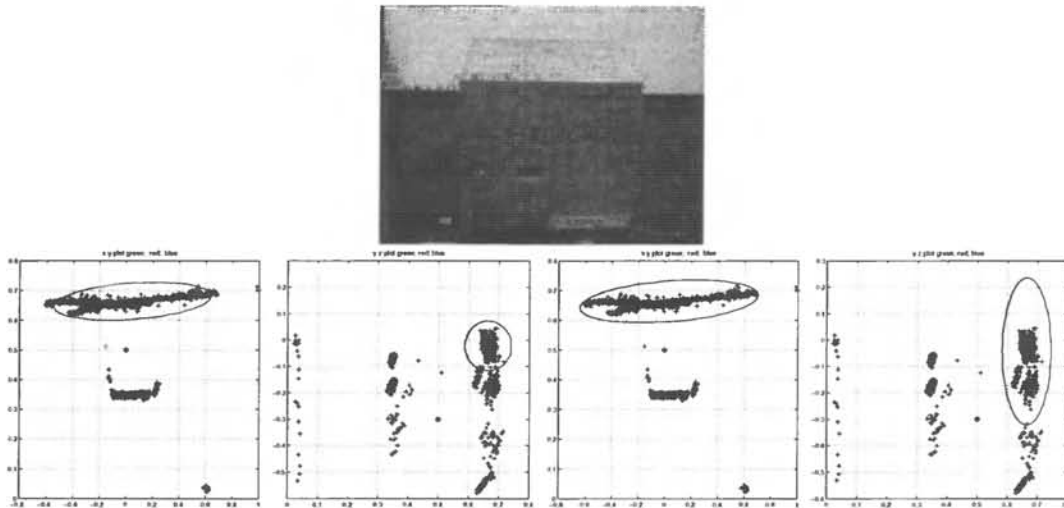

Figure 4: Example of the action of the priors on $a_i$ and $b_i$. The photograph shows a visual image of the scene: a box in front of a wall, and the priors were chosen to prefer a distribution matching the wall. The two left hand graphs show the top and side view of the scene clustered without priors, while the two right hand graphs use priors on $a_i$ and $b_i$. The priors give a preference for large values of $a_i$ and $b_i$, so biasing the optimization to find a mixture component matching the whole wall as opposed to just the top of it.

## 4   Classification and diagnosis

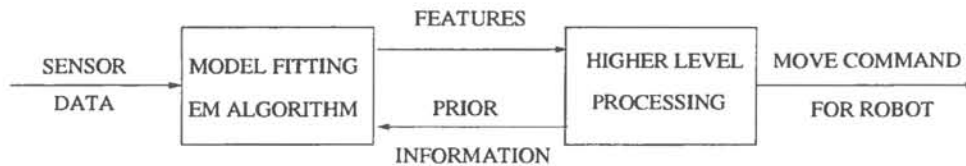

Figure 5: Schematic of system

This section describes how higher-level processing can be used to not only interpret the clusters fitted by the EM algorithm, but also affect the model-fitting using prior information. The processes of model-fitting and analysis are thus coupled, and not sequential.

The results of the model fitting are primarily processed to steer the robot. Once the cluster has been recognized as a box/wall/etc., the location and orientation are used to calculate a move command. To perform the object-recognition, we used a simple classifier on a feature vector extracted from the clustered data. The labels used were specific to docking, and commonly clustered objects – boxes, walls, thin vertical lines, but also included labels for clustering errors (like those shown in Figure 3). The features used were the values of the parameters $a_i$, $b_i$, giving the size of the clusters, but also measures of the visibility of the clusters, and the skewness of the within-cluster data. The classification used simple models of the probability distributions of the features $f_i$, given the objects $O_j$ (i.e. $P(f_i|O_j)$), using a set of training data. In addition to moving the robot, the classifier can modify the behavior of the model fitting algorithm. If a poor clustering solution is found, EM can be re-run with slightly different initial conditions. If the probable locations or sizes of objects are known from previous scans, or indeed from other sensors, then these can constrain the clustering through priors, or provide initial means.

## 5 Summary

This paper shows that the Mixture of Gaussians architecture combined with EM optimization and the use of parameter priors can be used to segment and analyze real data from the 3D range-finder of a mobile robot. The approach was successfully used to guide a mobile robot towards a docking object, using only its range-finder for perception.

For the learning community this provides more than an example of the application of a probabilistic model to a real task. We have shown how the usual Mixture of Gaussians model can be parameterized to include expectations about the environment in a way which can be readily extended. We have included prior knowledge at three different levels: 1. The use of problem-specific parameterization of the covariance matrix to find expected patterns (e.g. planes at particular angles). 2. The use of problem-specific parameter priors to automatically rule-out unlikely objects at the lowest level of perception. 3. The results of the clustering process were post-processed by higher-level classification algorithms which interpreted the parameters of the mixture components, diagnosed typical misclassification, provided new priors for future perception, and gave the robot control system new targets.

It is expected that the basic approach can be fruitfully applied to other sensors, to problems which track dynamically changing scenes, or to problems which require relationships between objects in the scene to be accounted for and interpreted. A problem common to all modeling approaches is that it is not trivial to determine the number and types of clusters needed to represent a given scene. Recent work with Markov-Chain Monte-Carlo approaches has been successfully applied to mixtures of Gaussians (Richardson and Green, 1997), allowing a Bayesian solution to this problem, which could provide control systems with even richer probabilistic information (a series of models conditioned on number of clusters).

### Acknowledgements

All authors were employed by Daimler-Benz AG during stages of the work. R. Murray-Smith gratefully acknowledges the support of Marie Curie TMR grant FMBICT961369.

## Footnotes

[1] By experimentation, a value of $\gamma$ of 0.01 was found to be reasonable for this application.

[2] Intuition for the $\hat{\theta}_i$ update can be obtained by considering that $(\mathbf{x}_{n1} - \mu_{i1})$ is the $x$ component of the distance between $\mathbf{x}_n$ and $\mu_i$, which is $|\mathbf{x}_n - \mu_i| \cos\theta$, and similarly $(\mathbf{x}_{n2} - \mu_{i2})$ is $|\mathbf{x}_n - \mu_i| \sin\theta$, so $\tan 2\theta = \frac{\sin 2\theta}{\cos 2\theta} = \frac{2\sin\theta\cos\theta}{\cos^2\theta - \sin^2\theta} = \frac{2(\mathbf{x}_{n1} - \mu_{i1})(\mathbf{x}_{n2} - \mu_{i2})}{(\mathbf{x}_{n1} - \mu_{i1})^2 - (\mathbf{x}_{n2} - \mu_{i2})^2}$.

[3]In this paper we make little use of the $\alpha$ priors, but introducing separate $\alpha_i$'s for each object could be a useful next step for scenes with varying object sizes.

[4]To deal with the case when $a_i = 0$, the prior is truncated, setting $p(a_i|\overline{a_i}) = 0$ when $a_i < \rho_{crit}$.

## References

Arman, F. and Aggarwal, J. K. (1993). Model-based object recognition in dense-range images—a review. *ACM Computing Surveys*, **25** (1), 5–43.

Dempster, A. P., Laird, N. M., and Rubin, D. B. (1977). Maximum likelihood from incomplete data via the EM algorithm. *J. Royal Statistical Society Series B*, **39**, 1–38.

Duda, R. O. and Hart, P. E. (1973). *Pattern Classification and Scene Analysis*. New York, Wiley.

McMichael, D. W. (1995). Bayesian growing and pruning strategies for MAP-optimal estimation of gaussian mixture models. In *4th IEE International Conf. on Artificial Neural Networks*, pp. 364–368.

PIAP (1995). PIAP impact report on TRC lidar performance. Technical Report 1, Industrial Research Institute for Automation and Measure ments, 02-486 Warszawa, Al. Jerozolimskie 202, Poland.

Richardson, S. and Green, P. J. (1997). On Bayesian anaysis of mixtures with an unknown number of components. *Journal of the Royal Statistical Society B*, **50** (4), 700–792.

Titterington, D., Smith, A., and Makov, U. (1985). *Statistical Analysis of Finite Mixture Distributions*. Chichester, John Wiley & Sons.
